# Annealing and the Rate Distortion Problem

**Albert E. Parker**
Department of Mathematical Sciences
Montana State University
Bozeman, MT 59771
parker@math.montana.edu

**Tomáš Gedeon**
Department of Mathematical Sciences
Montana State University
gedeon@math.montana.edu

**Alexander G. Dimitrov**
Center for Computational Biology
Montana State University
alex@nervana.montana.edu

## Abstract

In this paper we introduce methodology to determine the bifurcation structure of optima for a class of similar cost functions from Rate Distortion Theory, Deterministic Annealing, Information Distortion and the Information Bottleneck Method. We also introduce a numerical algorithm which uses the explicit form of the bifurcating branches to find optima at a bifurcation point.

## 1 Introduction

This paper analyzes a class of optimization problems

$$\max_{q \in \Delta} G(q) + \beta D(q) \tag{1}$$

where $\Delta$ is a linear constraint space, $G$ and $D$ are continuous, real valued functions of $q$, smooth in the interior of $\Delta$, and $\max_{q \in \Delta} G(q)$ is known. Furthermore, $G$ and $D$ are invariant under the group of symmetries $S_N$. The goal is to solve (1) for $\beta = \mathcal{B} \in [0, \infty)$.

This type of problem, which appears to be $NP$ hard, arises in Rate Distortion Theory [1, 2], Deterministic Annealing [3], Information Distortion [4, 5, 6] and the Information Bottleneck Method [7, 8].

The following basic algorithm, various forms of which have appeared in [3, 4, 6, 7, 8], can be used to solve (1) for $\beta = \mathcal{B}$.

**Algorithm 1** *Let*

$$q_0 \text{ be the maximizer of } \max_{q \in \Delta} G(q) \tag{2}$$

*and let $\beta_0 = 0$. For $k \geq 0$, let $(q_k, \beta_k)$ be a solution to (1). Iterate the following steps until $\beta_\kappa = \mathcal{B}$ for some $\kappa$.*

    *1. Perform $\beta$-step: Let $\beta_{k+1} = \beta_k + d_k$ where $d_k > 0$.*

2. *Take $q_{k+1}^{(0)} = q_k + \eta$, where $\eta$ is a small perturbation, as an initial guess for the solution $q_{k+1}$ at $\beta_{k+1}$.*

3. *Optimization: solve*

$$\max_{q \in \Delta} G(q) + \beta_{k+1} D(q)$$

*to get the maximizer $q_{k+1}$, using initial guess $q_{k+1}^{(0)}$.*

We introduce methodology to efficiently perform algorithm 1. Specifically, we implement numerical continuation techniques [9, 10] to effect steps 1 and 2. We show how to detect bifurcation and we rely on bifurcation theory with symmetries [11, 12, 13] to search for the desired solution branch. This paper concludes with the improved algorithm 6 which solves (1).

## 2 The cost functions

The four problems we analyze are from Rate Distortion Theory [1, 2], Deterministic Annealing [3], Information Distortion [4, 5, 6] and the Information Bottleneck Method [7, 8]. We discuss the explicit form of the cost function (i.e. $G(q)$ and $D(q)$) for each of these scenarios in this section.

### 2.1 The distortion function $D(q)$

Rate distortion theory is the information theoretic approach to the study of optimal source coding systems, including systems for quantization and data compression [2]. To define how well a source, the random variable $Y$, is represented by a particular representation using $N$ symbols, which we call $Y_N$, one introduces a *distortion function* between $Y$ and $Y_N$

$$D(q(y_N|y)) = D(Y, Y_N) = E_{y,y_N} d(y, y_N) = \sum_y \sum_{y_N} q(y_N|y) p(y) d(y, y_N)$$

where $d(y, y_N)$ is the *pointwise distortion function* on the individual elements of $y \in Y$ and $y_N \in Y_N$. $q(y_N|y)$ is a stochastic map or *quantization* of $Y$ into a representation $Y_N$ [1, 2]. The constraint space

$$\Delta := \{ q(y_N|y) \mid \sum_{y_N} q(y_N|y) = 1 \text{ and } q(y_N|y) \geq 0 \ \forall y \in Y \} \tag{3}$$

(compare with (1)) is the space of valid quantizers in $\Re^n$. A representation $Y_N$ is optimal if there is a quantizer $q^*(y_N|y)$ such that $D(q^*) = \min_{q \in \Delta} D(q)$.

In engineering and imaging applications, the distortion function is usually chosen as the *mean squared error* [1, 3, 14], $\hat{D}(Y, Y_N) = E_{y,y_N} \hat{d}(y, y_N)$, where the pointwise distortion function $\hat{d}(y, y_N)$ is the Euclidean squared distance. In this case, $\hat{D}(Y, Y_N)$ is a linear function of the quantizer. In [4, 5, 6], the *information distortion measure*

$$D_I(Y, Y_N) := \sum_{y,y_N} p(y, y_N) KL(p(x|y_N)||p(x|y)) = I(X; Y) - I(X; Y_N)$$

is used, where the Kullback-Leibler divergence $KL$ is the pointwise distortion function. Unlike the pointwise distortion functions usually investigated in information theory [1, 3], this one is nonlinear, it explicitly considers a third space, $X$, of inputs, and it depends on the quantizer $q(y_N|y)$ through $p(x|y_N) = \sum_y p(x|y) \frac{q(y_N|y)p(y)}{p(y_N)}$. The only term in $D_I$ which depends on the quantizer is $I(X; Y_N)$, so we can replace $D_I$ with the effective distortion

$$D_{eff}(q) := I(X; Y_N).$$

$D_{eff}(q)$ is the function $D(q)$ from (1) which has been considered in [4, 5, 6, 7, 8].

## 2.2 Rate Distortion

There are two related methods used to analyze communication systems at a distortion $D(q) \leq D_0$ for some given $D_0 \geq 0$ [1, 2, 3]. In rate distortion theory [1, 2], the problem of finding a minimum rate at a given distortion is posed as a *minimal information rate* distortion problem:

$$R(D_0) = \begin{array}{c} \min_{q(y_N|y) \in \Delta} I(Y;Y_N) \\ D(Y;Y_N) \leq D_0 \end{array} . \tag{4}$$

This formulation is justified by the Rate Distortion Theorem [1]. A similar exposition using the Deterministic Annealing approach [3] is a *maximal entropy* problem

$$\begin{array}{c} \max_{q(y_N|y) \in \Delta} H(Y_N|Y) \\ D(Y;Y_N) \leq D_0 \end{array} . \tag{5}$$

The justification for using (5) is Jayne's maximum entropy principle [15]. These formulations are related since $I(Y;Y_N) = H(Y_N) - H(Y_N|Y)$.

Let $I_0 > 0$ be some given information rate. In [4, 6], the neural coding problem is formulated as an entropy problem as in (5)

$$\begin{array}{c} \max_{q(y_N|y) \in \Delta} H(Y_N|Y) \\ D_{eff}(q) \geq I_0 \end{array} . \tag{6}$$

which uses the nonlinear effective information distortion measure $D_{eff}$.

Tishby et. al. [7, 8] use the information distortion measure to pose an information rate distortion problem as in (4)

$$\begin{array}{c} \min_{q(y_N|y) \in \Delta} I(Y;Y_N) \\ D_{eff}(q) \geq I_0 \end{array} . \tag{7}$$

Using the method of Lagrange multipliers, the rate distortion problems (4),(5),(6),(7) can be reformulated as finding the maxima of

$$\max_{q \in \Delta} F(q, \beta) = \max_{q \in \Delta}[G(q) + \beta D(q)] \tag{8}$$

as in (1) where $\beta = \mathcal{B}$. For the maximal entropy problem (6),

$$F(q, \beta) = H(Y_N|Y) + \beta D_{eff}(q) \tag{9}$$

and so $G(q)$ from (1) is the conditional entropy $H(Y_N|Y)$. For the minimal information rate distortion problem (7),

$$F(q, \beta) = -I(Y;Y_N) + \beta D_{eff}(q) \tag{10}$$

and so $G(q) = -I(Y;Y_N)$.

In [3, 4, 6], one explicitly considers $\mathcal{B} = \infty$. For (9), this involves taking $\lim_{\beta \to \infty} \max_{q \in \Delta} F(q, \beta) = \max_{q \in \Delta} D_{eff}(q)$ which in turn gives $\min_{q(y_N|y) \in \Delta} D_I$. In Rate Distortion Theory and the Information Bottleneck Method, one could be interested in solutions to (8) for finite $\mathcal{B}$ which takes into account a tradeoff between $I(Y;Y_N)$ and $D_{eff}$.

For lack of space, here we consider (9) and (10). Our analysis extends easily to similar formulations which use a norm based distortion such as $\hat{D}(q)$, as in [3].

## 3 Improving the algorithm

We now turn our attention back to algorithm 1 and indicate how numerical continuation [9, 10], and bifurcation theory with symmetries [11, 12, 13] can improve upon the choice of the algorithm's parameters.

We begin by rewriting (8), now incorporating the Lagrange multipliers for the equality constraint $\sum_{y_N} q(y_N | y_k) = 1$ from (3) which must be satisfied for each $y_k \in Y$. This gives the Lagrangian

$$\mathcal{L}(q, \lambda, \beta) = F(q, \beta) + \sum_{k=1}^{K} \lambda_k \left( \sum_{y_N} q(y_N | y_k) - 1 \right). \tag{11}$$

There are optimization schemes, such as the Fixed Point [4, 6] and projected Augmented Lagrangian [6, 16] methods, which exploit the structure of (11) to find local solutions to (8) for step 3 of algorithm 1.

## 3.1 Bifurcation structure of solutions

It has been observed that the solutions $\{q_k\}$ undergo *bifurcations* or *phase transitions* [3, 4, 6, 7, 8]. We wish to pose (8) as a dynamical system in order to study the *bifurcation structure* of local solutions for $\beta \in [0, \mathcal{B}]$. To this end, consider the equilibria of the flow

$$\begin{pmatrix} \dot{q} \\ \dot{\lambda} \end{pmatrix} = \nabla_{q,\lambda} \mathcal{L}(q, \lambda, \beta) \tag{12}$$

for $\beta \in [0, \mathcal{B}]$. These are points $\begin{pmatrix} q^* \\ \lambda^* \end{pmatrix}$ where $\nabla_{q,\lambda} \mathcal{L}(q^*, \lambda^*, \beta) = 0$ for some $\beta$. The Jacobian of this system is the Hessian $\Delta_{q,\lambda} \mathcal{L}(q, \lambda, \beta)$. Equilibria, $(q^*, \lambda^*)$, of (12), for which $\Delta_q F(q^*, \beta)$ is negative definite, are local solutions of (8) [16, 17].

Let $|Y| = K$, $|Y_N| = N$, and $n = NK$. Thus, $q \in \Delta \subset \Re^n$ and $\lambda \in \Re^K$. The $(n + K) \times (n + K)$ Hessian of (11) is

$$\Delta_{q,\lambda} \mathcal{L}(q, \lambda, \beta) = \begin{pmatrix} \Delta_q F(q, \beta) & J^T \\ J & \mathbf{0} \end{pmatrix}$$

where $\mathbf{0}$ is $K \times K$ [17]. $\Delta_q F$ is the $n \times n$ block diagonal matrix of $N$ $K \times K$ matrices $\{B_i\}_{i=1}^{N}$ [4]. $J$ is the $K \times n$ Jacobian of the vector of $K$ constraints from (11),

$$J = \underbrace{\begin{pmatrix} I_K & I_K & ... & I_K \end{pmatrix}}_{N \text{ blocks}}. \tag{13}$$

The kernel of $\Delta_{q,\lambda} \mathcal{L}$ plays a pivotal role in determining the bifurcation structure of solutions to (8). This is due to the fact that bifurcation of an equilibria $(q^*, \lambda^*)$ of (12) at $\beta = \beta^*$ happen when $\ker \Delta_{q,\lambda} \mathcal{L}(q^*, \lambda^*, \beta^*)$ is nontrivial. Furthermore, the bifurcating branches are tangent to certain linear subspaces of $\ker \Delta_{q,\lambda} \mathcal{L}(q^*, \lambda^*, \beta^*)$ [12].

## 3.2 Bifurcations with symmetry

Any solution $q^*(y_N | y)$ to (8) gives another equivalent solution simply by permuting the labels of the classes of $Y_N$. For example, if $P_1$ and $P_2$ are two $n \times 1$ vectors such that for a solution $q^*(y_N | y)$, $q^*(y_N = 1 | y) = P_1$ and $q^*(y_N = 2 | y) = P_2$, then the quantizer where $\hat{q}(y_N = 1 | y) = P_2$, $\hat{q}(y_N = 2 | y) = P_1$ and $\hat{q}(y_N | y) = q^*(y_N | y)$ for all other classes $y_N$ is a maximizer of (8) with $F(\hat{q}, \beta) = F(q^*, \beta)$. Let $S_N$ be the algebraic group of all permutations on $N$ symbols [18, 19]. We say that $F(q, \beta)$ is $S_N$-*invariant* if $F(q, \beta) = F(\sigma(q), \beta)$ where $\sigma(q)$ denotes the action on $q$ by permutation of the classes of $Y_N$ as defined by any $\sigma \in S_N$ [17]. Now suppose that a solution $q^*$ is fixed by all the elements of $S_M$ for $M \leq N$. Bifurcations at $\beta = \beta^*$ in this scenario are called *symmetry breaking* if the bifurcating solutions are fixed (and only fixed) by subgroups of $S_M$.

To determine where a bifurcation of a solution $(q^*, \lambda^*, \beta)$ occurs, one determines $\beta$ for which $\Delta_q F(q^*, \beta)$ has a nontrivial kernel. This approach is justified by the fact that $\Delta_{q,\lambda} \mathcal{L}(q^*, \lambda^*, \beta)$ is singular if and only if $\Delta_q F(q^*, \beta)$ is singular [17]. At a bifurcation $(q^*, \lambda^*, \beta^*)$ where $q^*$ is fixed by $S_M$ for $M \leq N$, $\Delta_q F(q^*, \beta^*)$ has $M$ identical blocks. The bifurcation is generic if

$$\text{each of the identical blocks has a single 0-eigenvector, } \boldsymbol{v}, \text{ and the other blocks are nonsingular.} \tag{14}$$

Thus, a generic bifurcation can be detected by looking for singularity of one of the $K \times K$ identical blocks of $\Delta_q F(q^*, \beta)$. We call the classes of $Y_N$ which correspond to identical blocks *unresolved* classes. The classes of $Y_N$ that are not unresolved are called *resolved* classes.

The Equivariant Branching Lemma and the Smoller-Wasserman Theorem [12, 13] ascertain the existence of explicit bifurcating solutions in subspaces of $\ker \Delta_{q,\lambda} \mathcal{L}(q^*, \lambda^*, \beta^*)$ which are fixed by special subgroups of $S_M$ [12, 13]. Of particular interest are the bifurcating solutions in subspaces of $\ker \Delta_{q,\lambda} \mathcal{L}(q^*, \lambda^*, \beta^*)$ of dimension 1 guaranteed by the following theorem

**Theorem 2** *[17] Let $(q^*, \lambda^*, \beta^*)$ be a generic bifurcation of (12) which is fixed (and only fixed) by $S_M$, for $1 < M \leq N$. Then, for small t, with $\beta(t = 0) = \beta^*$, there exists $M$ bifurcating solutions,*

$$\begin{pmatrix} q^* \\ \lambda^* \\ \beta^* \end{pmatrix} + \begin{pmatrix} t\boldsymbol{u}_m \\ \beta(t) \end{pmatrix}, \text{ where } 1 \leq m \leq M, \tag{15}$$

$$[\boldsymbol{u}_m]_\nu = \begin{cases} (M-1)\boldsymbol{v} & \text{if } \nu \text{ is the } m^{th} \text{ unresolved class of } Y_N \\ -\boldsymbol{v} & \text{if } \nu \text{ is some other unresolved class of } Y_N \\ \boldsymbol{0} & \text{otherwise} \end{cases} \tag{16}$$

*and $\boldsymbol{v}$ is defined as in (14). Furthermore, each of these solutions is fixed by the symmetry group $S_{M-1}$.*

For a bifurcation from the uniform quantizer, $q_{\frac{1}{N}}$, which is identically $\frac{1}{N}$ for all $y$ and all $y_N$, all of the classes of $Y_N$ are unresolved. In this case,

$$\boldsymbol{u}_m = (-\boldsymbol{v}^T, ..., -\boldsymbol{v}^T, (N-1)\boldsymbol{v}^T, -\boldsymbol{v}^T, ..., -\boldsymbol{v}^T, \boldsymbol{0}^T)^T$$

where $(N-1)\boldsymbol{v}$ is in the $m^{th}$ component of $\boldsymbol{u}_m$.

Relevant to the computationalist is that instead of looking for a bifurcation by looking for singularity of the $n \times n$ Hessian $\Delta_q F(q^*, \beta)$, one may look for singularity of one of the $K \times K$ identical blocks, where $K = \frac{n}{N}$. After bifurcation of a local solution to (8) has been detected at $\beta = \beta^*$, knowledge of the bifurcating directions makes finding solutions of interest for $\beta > \beta^*$ much easier (see section 3.4.1).

### 3.3 The subcritical bifurcation

In all problems under consideration, the solution for $\beta = 0$ is known. For (9), (10) this solution is $q_0 = q_{\frac{1}{N}}$. For (4) and (5), $q_0$ is the mean of $Y$. Rose [3] was able to compute explicitly the critical value $\beta^*$ where $q_0$ loses stability for the Euclidean pointwise distortion function. We have the following related result.

**Theorem 3** *[20] Consider problems (9), (10). The solution $q_0 = 1/N$ loses stability at $\beta = \beta^*$ where $1/\beta^*$ is the second largest eigenvalue of a discrete Markov chain on vertices $y \in Y$, where the transition probabilities $p(y_l \to y_k) := \sum_i p(y_k|x_i)p(x_i|y_l)$.*

**Corollary 4** *Bifurcation of the solution* $(q_{\frac{1}{N}}, \beta)$ *in (9), (10) occurs at* $\beta \geq 1$.

The *discriminant* of the bifurcating branch (15) is defined as [17]

$$
\begin{aligned}
\zeta(q^*, \beta^*, \boldsymbol{u}_m) &= \langle \boldsymbol{u}_m, \partial_{q,\lambda}^3 \mathcal{L}(q^*, \lambda^*, \beta^*)[\boldsymbol{u}_m, EL^- E \partial_{q,\lambda}^3 \mathcal{L}(q^*, \lambda^*, \beta^*)[\boldsymbol{u}_m, \boldsymbol{u}_m]] \rangle \\
&\quad -3 \langle \boldsymbol{u}_m, \partial_{q,\lambda}^4 \mathcal{L}(q^*, \lambda^*, \beta^*)[\boldsymbol{u}_m, \boldsymbol{u}_m, \boldsymbol{u}_m] \rangle,
\end{aligned}
$$

where $\langle \cdot, \cdot \rangle$ is the Euclidean inner product, $\partial_{q,\lambda}^n \mathcal{L}[\cdot, ..., \cdot]$ is the multilinear form of the $n^{th}$ derivative of $\mathcal{L}$, $E$ is the projection matrix onto $\mathrm{range}(\Delta_{q,\lambda} \mathcal{L}(q^*, \lambda^*, \beta^*))$, and $L^-$ is the Moore-Penrose generalized inverse of the Hessian $\Delta_{q,\lambda} \mathcal{L}(q^*, \lambda^*, \beta^*)$.

**Theorem 5** *[17] If* $\zeta(q^*, \beta^*, \boldsymbol{u}_m) < 0$*, then the bifurcating branch (15) is subcritical (i.e. a first order phase transition). If* $\zeta(q^*, \beta^*, \boldsymbol{u}_m) > 0$*, then (15) is supercritical.*

For a data set with a joint probability distribution modelled by a mixture of four Gaussians as in [4], Theorem 5 predicts a subcritical bifurcation from $(q_{\frac{1}{N}}, \beta^* \approx 1.038706)$ for (9) when $N \geq 3$. The existence of a subcritical bifurcation (a first order phase transition) is intriguing.

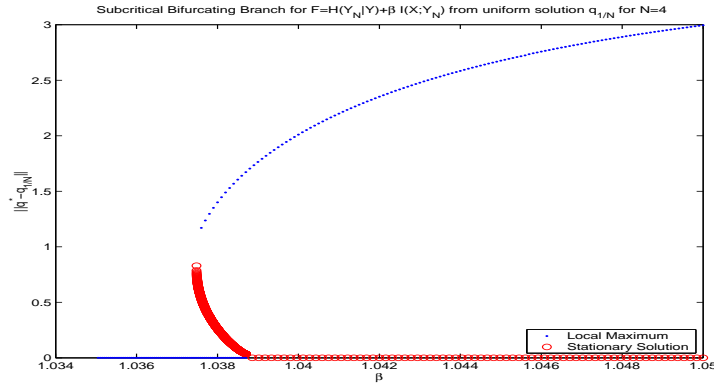

Figure 1: A joint probability space on the random variables $(X, Y)$ was constructed from a mixture of four Gaussians as in [4]. Using this probability space, the equilibria of (12) for $F$ as defined in (9) were found using Newton's method. Depicted is the subcritical bifurcation from $(q_{\frac{1}{4}}, \beta^* \approx 1.038706)$.

In analogy to the rate distortion curve [2, 1], we can define an $H$-$I$ curve for the problem (6)

$$
H(I_0) := \max_{q \in \Delta, D_{eff} \geq I_0} H(Y_N | Y).
$$

Let $I_{\max} = \max_{q \in \Delta} D_{eff}$. Then for each $I_0 \in (0, I_{\max})$ the value $H(I_0)$ is well defined and achieved at a point where $D_{eff} = I_0$. At such a point there is a Lagrange multiplier $\beta$ such that $\nabla_{q,\lambda} \mathcal{L} = \mathbf{0}$ (compare with (11) and (12)) and this $\beta$ solves problem (9). Therefore, for each $I \in (0, I_{\max})$, there is a corresponding $\beta$ which solves problem (9). The existence of a subcritical bifurcation in $\beta$ implies that this correspondence is not monotone for small values of $I$.

### 3.4 Numerical Continuation

Numerical *continuation* methods efficiently analyze the solution behavior of dynamical systems such as (12) [9, 10]. Continuation methods can speed up the search for the solution $q_{k+1}$ at $\beta_{k+1}$ in step 3 of algorithm 1 by improving upon the perturbed choice $q_{k+1}^{(0)} = q_k + \eta$. First,

the vector $(\partial_\beta q_k^T \; \partial_\beta \lambda_k^T)^T$ which is tangent to the curve $\nabla_{q,\lambda} \mathcal{L}(q, \lambda, \beta) = \mathbf{0}$ at $(q_k, \lambda_k, \beta_k)$ is computed by solving the matrix system

$$\Delta_{q,\lambda} \mathcal{L}(q_k, \lambda_k, \beta_k) \begin{pmatrix} \partial_\beta q_k \\ \partial_\beta \lambda_k \end{pmatrix} = -\partial_\beta \nabla_{q,\lambda} \mathcal{L}(q_k, \lambda_k, \beta_k). \qquad (17)$$

Now the initial guess in step 2 becomes $q_{k+1}^{(0)} = q_k + d_k \partial_\beta q_k$ where $d_k = \frac{\Delta s}{\sqrt{||\partial_\beta q_k||^2 + ||\partial_\beta \lambda_k||^2 + 1}}$ for $\Delta s > 0$. Furthermore, $\beta_{k+1}$ in step 1 is found by using this same $d_k$. This choice of $d_k$ assures that a fixed step along $(\partial_\beta q_k^T \; \partial_\beta \lambda_k^T)^T$ is taken for each $k$. We use three different continuation methods which implement variations of this scheme: *Parameter, Tangent* and *Pseudo Arc-Length* [9, 17]. These methods can greatly decrease the optimization iterations needed to find $q_{k+1}$ from $q_{k+1}^{(0)}$ in step 3. The cost savings can be significant, especially when continuation is used in conjunction with a Newton type optimization scheme which explicitly uses the Hessian $\Delta_q F(q_k, \beta_k)$. Otherwise, the CPU time incurred from solving (17) may outweigh this benefit.

### 3.4.1 Branch switching

Suppose that a bifurcation of a solution $q^*$ of (8) has been detected at $\beta^*$. To proceed, one uses the explicit form of the bifurcating directions, $\{\mathbf{u}_m\}_{m=1}^M$ from (16) to search for the bifurcating solution of interest, say $q_{k+1}$, whose existence is guaranteed by Theorem 2. To do this, let $\mathbf{u} = \mathbf{u}_m$ for some $m \le M$, then implement a *branch switch* [9]

$$q_{k+1}^{(0)} = q^* + d_k \cdot \mathbf{u}.$$

## 4 A numerical algorithm

We conclude with a numerical algorithm to solve (1). The section numbers in parentheses indicate the location in the text supporting each step.

**Algorithm 6** *Let $q_0$ be the maximizer of $\max_{q \in \Delta} G$, $\beta_0 = 1$ (3.3) and $\Delta s > 0$. For $k \ge 0$, let $(q_k, \beta_k)$ be a solution to (1). Iterate the following steps until $\beta_\kappa = \mathcal{B}$ for some $\kappa$.*

1. *(3.4) Perform $\beta$-step: solve (17) for $(\partial_\beta q_k^T \; \partial_\beta \lambda_k^T)^T$ and select $\beta_{k+1} = \beta_k + d_k$ where $d_k = \frac{\Delta s}{\sqrt{||\partial_\beta q_k||^2 + ||\partial_\beta \lambda_k||^2 + 1}}$.*

2. *(3.4) The initial guess for $q_{k+1}$ at $\beta_{k+1}$ is $q_{k+1}^{(0)} = q_k + d_k \cdot \partial_\beta q_k$.*

3. *Optimization: solve*

$$\max_{q \in \Delta} G(q) + \beta_{k+1} D(q)$$

   *to get the maximizer $q_{k+1}$, using initial guess $q_{k+1}^{(0)}$.*

4. *(3.2) Check for bifurcation: compare the sign of the determinant of an identical block of each of*

$$\Delta_q[G(q_k) + \beta_k D(q_k)] \text{ and } \Delta_q[G(q_{k+1}) + \beta_{k+1} D(q_{k+1})].$$

   *If a bifurcation is detected, then set $q_{k+1}^{(0)} = q_k + d_k \cdot \mathbf{u}$ where $\mathbf{u}$ is defined as in (16) for some $m \le M$, and repeat step 3.*

### Acknowledgments

Many thanks to Dr. John P. Miller at the Center for Computational Biology at Montana State University-Bozeman. This research is partially supported by NSF grants DGE 9972824, MRI 9871191, and EIA-0129895; and NIH Grant R01 MH57179.

# References

[1] Thomas Cover and Jay Thomas. *Elements of Information Theory*. Wiley Series in Communication, New York, 1991.

[2] Robert M. Gray. *Entropy and Information Theory*. Springer-Verlag, 1990.

[3] Kenneth Rose. Deteministic annealing for clustering, compression, classification, regerssion, and related optimization problems. *Proc. IEEE*, 86(11):2210–2239, 1998.

[4] Alexander G. Dimitrov and John P. Miller. Neural coding and decoding: communication channels and quantization. *Network: Computation in Neural Systems*, 12(4):441–472, 2001.

[5] Alexander G. Dimitrov and John P. Miller. Analyzing sensory systems with the information distortion function. In Russ B Altman, editor, *Pacific Symposium on Biocomputing 2001*. World Scientific Publushing Co., 2000.

[6] Tomas Gedeon, Albert E. Parker, and Alexander G. Dimitrov. Information distortion and neural coding. *Canadian Applied Mathematics Quarterly*, 2002.

[7] Naftali Tishby, Fernando C. Pereira, and William Bialek. The information bottleneck method. The 37th annual Allerton Conference on Communication, Control, and Computing, 1999.

[8] Noam Slonim and Naftali Tishby. Agglomerative information bottleneck. In S. A. Solla, T. K. Leen, and K.-R. Müller, editors, *Advances in Neural Information Processing Systems*, volume 12, pages 617–623. MIT Press, 2000.

[9] Wolf-Jurgen Beyn, Alan Champneys, Eusebius Doedel, Willy Govaerts, Yuri A. Kuznetsov, and Bjorn Sandstede. *Handbook of Dynamical Systems III*. World Scientific, 1999. Chapter in book: Numerical Continuation and Computation of Normal Forms.

[10] Eusebius Doedel, Herbert B. Keller, and Jean P. Kernevez. Numerical analysis and control of bifurcation problems in finite dimensions. *International Journal of Bifurcation and Chaos*, 1:493–520, 1991.

[11] M. Golubitsky and D. G. Schaeffer. *Singularities and Groups in Bifurcation Theory I*. Springer Verlag, New York, 1985.

[12] M. Golubitsky, I. Stewart, and D. G. Schaeffer. *Singularities and Groups in Bifurcation Theory II*. Springer Verlag, New York, 1988.

[13] J. Smoller and A. G. Wasserman. Bifurcation and symmetry breaking. *Inventiones mathematicae*, 100:63–95, 1990.

[14] Allen Gersho and Robert M. Gray. *Vector Quantization and Signal Compression*. Kluwer Academic Publishers, 1992.

[15] E. T. Jaynes. On the rationale of maximum-entropy methods. *Proc. IEEE*, 70:939–952, 1982.

[16] J. Nocedal and S. J. Wright. *Numerical Optimization*. Springer, New York, 2000.

[17] Albert E. Parker III. Solving the rate distortion problem. PhD thesis, Montana State University, 2003.

[18] H. Boerner. *Representations of Groups*. Elsevier, New York, 1970.

[19] D. S. Dummit and R. M. Foote. *Abstract Algebra*. Prentice Hall, NJ, 1991.

[20] Tomas Gedeon and Bryan Roosien. Phase transitions in information distortion. In preparation, 2003.
